# Rules and Similarity in Concept Learning

**Joshua B. Tenenbaum**
Department of Psychology
Stanford University, Stanford, CA 94305
*jbt@psych.stanford.edu*

## Abstract

This paper argues that two apparently distinct modes of generalizing concepts – abstracting rules and computing similarity to exemplars – should both be seen as special cases of a more general Bayesian learning framework. Bayes explains the specific workings of these two modes – which rules are abstracted, how similarity is measured – as well as why generalization should appear rule- or similarity-based in different situations. This analysis also suggests why the rules/similarity distinction, even if not computationally fundamental, may still be useful at the algorithmic level as part of a principled approximation to fully Bayesian learning.

## 1 Introduction

In domains ranging from reasoning to language acquisition, a broad view is emerging of cognition as a hybrid of two distinct modes of computation, one based on applying abstract rules and the other based on assessing similarity to stored exemplars [7]. Much support for this view comes from the study of concepts and categorization. In generalizing concepts, people's judgments often seem to reflect both rule-based and similarity-based computations [9], and different brain systems are thought to be involved in each case [8]. Recent psychological models of classification typically incorporate some combination of rule-based and similarity-based modules [1,4]. In contrast to this currently popular modularity position, I will argue here that rules and similarity are best seen as two ends of a continuum of possible concept representations. In [11,12], I introduced a general theoretical framework to account for how people can learn concepts from just a few positive examples based on the principles of Bayesian inference. Here I explore how this framework provides a unifying explanation for these two apparently distinct modes of generalization. The Bayesian framework not only includes both rules and similarity as special cases but also addresses several questions that conventional modular accounts do not. People employ particular algorithms for selecting rules and measuring similarity. Why these algorithms as opposed to any others? People's generalizations appear to shift from similarity-like patterns to rule-like patterns in systematic ways, e.g., as the number of examples observed increases. Why these shifts?

This short paper focuses on a simple learning game involving number concepts, in which both rule-like and similarity-like generalizations clearly emerge in the judgments of human subjects. Imagine that I have written some short computer programs which take as input a natural number and return as output either "yes" or "no" according to whether that number

satisfies some simple concept. Some possible concepts might be "$x$ is odd", "$x$ is between 30 and 45", "$x$ is a power of 3", or "$x$ is less than 10". For simplicity, we assume that only numbers under 100 are under consideration. The learner is shown a few randomly chosen *positive* examples – numbers that the program says "yes" to – and must then identify the other numbers that the program would accept. This task, admittedly artificial, nonetheless draws on people's rich knowledge of number while remaining amenable to theoretical analysis. Its structure is meant to parallel more natural tasks, such as word learning, that often require meaningful generalizations from only a few positive examples of a concept.

Section 2 presents representative experimental data for this task. Section 3 describes a Bayesian model and contrasts its predictions with those of models based purely on rules or similarity. Section 4 summarizes and discusses the model's applicability to other domains.

## 2   The number concept game

Eight subjects participated in an experimental study of number concept learning, under essentially the same instructions as those given above [11]. On each trial, subjects were shown one or more random positive examples of a concept and asked to rate the probability that each of 30 test numbers would belong to the same concept as the examples observed. $X$ denotes the set of examples observed on a particular trial, and $n$ the number of examples.

Trials were designed to fall into one of three classes. Figure 1a presents data for two representative trials of each class. Bar heights represent the average judged probabilities that particular test numbers fall under the concept given one or more positive examples $X$, marked by "*"s. Bars are shown only for those test numbers rated by subjects; missing bars do *not* denote zero probability of generalization, merely missing data.

On class I trials, subjects saw only one example of each concept: e.g., $X = \{16\}$ and $X = \{60\}$. To minimize bias, these trials preceded all others on which multiple examples were given. Given only one example, people gave most test numbers fairly similar probabilities of acceptance. Numbers that were intuitively more similar to the example received slightly higher ratings: e.g., for $X = \{16\}$, 8 was more acceptable than 9 or 6, and 17 more than 87; for $X = \{60\}$, 50 was more acceptable than 51, and 63 more than 43.

The remaining trials each presented four examples and occured in pseudorandom order. On class II trials, the examples were consistent with a simple mathematical rule: $X = \{16, 8, 2, 64\}$ or $X = \{60, 80, 10, 30\}$. Note that the obvious rules, "powers of two" and "multiples of ten", are in no way logically implied by the data. "Multiples of five" is a possibility in the second case, and "even numbers" or "all numbers under 80" are possibilities in both, not to mention other logically possible but psychologically implausible candidates, such as "all powers of two, except 32 or 4". Nonetheless, subjects overwhelmingly followed an all-or-none pattern of generalization, with all test numbers rated near 0 or 1 according to whether they satisfied the single intuitively "correct" rule. These preferred rules can be loosely characterized as the *most specific* rules (i.e., with smallest extension) that include all the examples and that also meet some criterion of psychological simplicity.

On class III trials, the examples satisfied no simple mathematical rule but did have similar magnitudes: $X = \{16, 23, 19, 20\}$ and $X = \{60, 52, 57, 55\}$. Generalization now followed a similarity gradient along the dimension of magnitude. Probability ratings fell below 0.5 for numbers more than a characteristic distance $\xi$ beyond the largest or smallest observed examples – roughly the typical distance between neighboring examples ($\sim$ 2 or 3). Logically, there is no reason why participants could not have generalized according to

various complex rules that happened to pick out the given examples, or according to very different values of $\xi$, yet all subjects displayed more or less the same similarity gradients.

To summarize these data, generalization from a single example followed a weak similarity gradient based on both mathematical and magnitude properties of numbers. When several more examples were observed, generalization evolved into either an all-or-none pattern determined by the most specific simple rule, or, when no simple rule applied, a more articulated magnitude-based similarity gradient falling off with characteristic distance $\xi$ roughly equal to the typical separation between neighboring examples. Similar patterns were observed on several trials not shown (including one with a different value of $\xi$) and on two other experiments in quite different domains (described briefly in Section 4).

## 3   The Bayesian model

In [12], I introduced a Bayesian framework for concept learning in the context of learning axis-parallel rectangles in a multidimensional feature space. Here I show that the same framework can be adapted to the more complex situation of learning number concepts and can explain all of the phenomena of rules and similarity documented above. Formally, we observe $n$ positive examples $X = \{x^{(1)}, \ldots, x^{(n)}\}$ of concept $C$ and want to compute $p(y \in C|X)$, the probability that some new object $y$ belongs to $C$ given the observations $X$. Inductive leverage is provided by a hypothesis space $\mathcal{H}$ of possible concepts and a probabilistic model relating hypotheses $h$ to data $X$.

**The hypothesis space.** Elements of $\mathcal{H}$ correspond to subsets of the universe of objects that are psychologically plausible candidates for the extensions of concepts. Here the universe consists of numbers between 1 and 100, and the hypotheses correspond to subsets such as the even numbers, the numbers between 1 and 10, etc. The hypotheses can be thought of in terms of either rules or similarity, i.e., as potential rules to be abstracted or as features entering into a similarity computation, but Bayes does not distinguish these interpretations.

Because we can capture only a fraction of the hypotheses people might bring to this task, we would like an objective way to focus on the most relevant parts of people's hypothesis space. One such method is *additive clustering (ADCLUS)* [6,10], which extracts a set of features that best accounts for subjects' similarity judgments on a given set of objects. These features simply correspond to subsets of objects and are thus naturally identified with hypotheses for concept learning. Applications of ADCLUS to similarity judgments for the numbers 0-9 reveal two kinds of subsets [6,10]: numbers sharing a common mathematical property, such as $\{2, 4, 8\}$ and $\{3, 6, 9\}$, and consecutive numbers of similar magnitude, such as $\{1, 2, 3, 4\}$ and $\{2, 3, 4, 5, 6\}$. Applying ADCLUS to the full set of numbers from 1 to 100 is impractical, but we can construct an analogous hypothesis space for this domain based on the two kinds of hypotheses found in the ADCLUS solution for 0-9. One group of hypotheses captures salient mathematical properties: odd, even, square, cube, and prime numbers, multiples and powers of small numbers ($\leq 12$), and sets of numbers ending in the same digit. A second group of hypotheses, representing the dimension of numerical magnitude, includes all intervals of consecutive numbers with endpoints between 1 and 100.

**Priors and likelihoods.** The probabilistic model consists of a prior $p(h)$ over $\mathcal{H}$ and a likelihood $p(X|h)$ for each hypothesis $h \in H$. Rather than assigning prior probabilities to each of the 5083 hypotheses individually, I adopted a hierarchical approach based on the intuitive division of $\mathcal{H}$ into mathematical properties and magnitude intervals. A fraction $\lambda$ of the total probability was allocated to the mathematical hypotheses as a group, leaving $(1 - \lambda)$ for

the magnitude hypotheses. The $\lambda$ probability was distributed uniformly across the mathematical hypotheses. The $(1 - \lambda)$ probability was distributed across the magnitude intervals as a function of interval size according to an Erlang distribution, $p(h) \propto (|h|/\sigma^2)e^{-|h|/\sigma}$, to capture the intuition that intervals of some intermediate size are more likely than those of very large or small size. $\lambda$ and $\sigma$ are treated as free parameters of the model.

The likelihood is determined by the assumption of randomly sampled positive examples. In the simplest case, each example in $X$ is assumed to be independently sampled from a uniform density over the concept $C$. For $n$ examples we then have:

$$
\begin{aligned}
p(X|h) &= 1/|h|^n \ \text{ if } \ \forall j, x^{(j)} \in h \\
&= 0 \ \text{ otherwise,}
\end{aligned}
\tag{1}
$$

where $|h|$ denotes the size of the subset $h$. For example, if $h$ denotes the even numbers, then $|h| = 50$, because there are 50 even numbers between 1 and 100. Equation 1 embodies the *size principle* for scoring hypotheses: smaller hypotheses assign greater likelihood than do larger hypotheses to the same data, and they assign exponentially greater likelihood as the number of consistent examples increases. The size principle plays a key role in learning concepts from only positive examples [12], and, as we will see below, in determining the appearance of rule-like or similarity-like modes of generalization.

Given these priors and likelihoods, the posterior $p(h|X)$ follows directly from Bayes' rule. Finally, we compute the probability of generalization to a new object $y$ by averaging the predictions of all hypotheses weighted by their posterior probabilities $p(h|X)$:

$$
p(y \in C|X) = \sum_{h \in \mathcal{H}} p(y \in C|h)p(h|X).
\tag{2}
$$

Equation 2 follows from the conditional independence of $X$ and the membership of $y \in C$, given $h$. To evaluate Equation 2, note that $p(y \in C|h)$ is simply 1 if $y \in h$, and 0 otherwise.

**Model results.** Figure 1b shows the predictions of this Bayesian model (with $\lambda = 1/2$, $\sigma = 10$). The model captures the main features of the data, including convergence to the most specific rule on Class II trials and to appropriately shaped similarity gradients on Class III trials. We can understand the transitions between graded, similarity-like and all-or-none, rule-like regimes of generalization as arising from the interaction of the *size principle* (Equation 1) with *hypothesis averaging* (Equation 2). Because each hypothesis $h$ contributes to the average in Equation 2 in proportion to its posterior probability $p(h|X)$, the degree of uncertainty in $p(h|X)$ determines whether generalization will be sharp or graded. When $p(h|X)$ is very spread out, many distinct hypotheses contribute significantly, resulting in a broad gradient of generalization. When $p(h|X)$ is concentrated on a single hypothesis $h^*$, only $h^*$ contributes significantly and generalization appears all-or-none. The degree of uncertainty in $p(h|X)$ is in turn a consequence of the size principle. Given a few examples consistent with one hypothesis that is significantly smaller than the next-best competitor – such as $X = \{16, 8, 2, 64\}$, where "powers of two" is significantly smaller than "even numbers" – then the smallest hypothesis becomes exponentially more likely than any other and generalization appears to follow this most specific rule. However, given only one example (such as $X = \{16\}$), or given several examples consistent with many similarly sized hypotheses – such as $X = \{16, 23, 19, 20\}$, where the top candidates are all very similar intervals: "numbers between 16 and 23", "numbers between 15 and 24", etc. – the size-based likelihood favors the smaller hypotheses only slightly, $p(h|X)$ is spread out over many overlapping hypotheses and generalization appears to follow a gradient of similarity. That the Bayesian

model predicts the right shape for the magnitude-based similarity gradients on Class III trials is no accident. The characteristic distance $\xi$ of the Bayesian generalization gradient varies with the uncertainty in $p(h|X)$, which (for interval hypotheses) can be shown to covary with the intuitively relevant factor of average separation between neighboring examples.

**Bayes vs. rules or similarity alone.** It is instructive to consider two special cases of the Bayesian model that are equivalent to conventional similarity-based and rule-based algorithms from the concept learning literature. What I call the SIM algorithm was pioneered by [5] and also described in [2,3] as a Bayesian approach to learning concepts from both positive and negative evidence. SIM replaces the size-based likelihood with a binary likelihood that measures only whether a hypothesis is consistent with the examples: $p(X|h) = 1$ if $\forall j, x^{(j)} \in h$, and 0 otherwise. Generalization under SIM is just a count of the features shared by $y$ and all the examples in $X$, independent of the frequency of those features or the number of examples seen. As Figure 1c shows, SIM successfully models generalization from a single example (Class I) but fails to capture how generalization sharpens up after multiple examples, to either the most specific rule (Class II) or a magnitude-based similarity gradient with appropriate characteristic distance $\xi$ (Class III). What I call the MIN algorithm preserves the size principle but replaces the step of hypothesis averaging with maximization: $p(y \in C|X) = 1$ if $y \in \arg\max_h p(X|h)$, and 0 otherwise. MIN is perhaps the oldest algorithm for concept learning [3] and, as a maximum likelihood algorithm, is asymptotically equivalent to Bayes. Its success for finite amounts of data depends on how peaked $p(h|X)$ is (Figure 1d). MIN always selects the most specific consistent rule, which is reasonable when that hypothesis is much more probable than any other (Class II), but too conservative in other cases (Classes I and III). In quantitative terms, the predictions of Bayes correlate much more highly with the observed data ($R^2 = 0.91$) than do the predictions of either SIM ($R^2 = 0.74$) or MIN ($R^2 = 0.47$). In sum, only the full Bayesian framework can explain the full range of rule-like and similarity-like generalization patterns observed on this task.

## 4 Discussion

Experiments in two other domains provide further support for Bayes as a unifying framework for concept learning. In the context of multidimensional continuous feature spaces, similarity gradients are the default mode of generalization [5]. Bayes successfully models how the shape of those gradients depends on the distribution and number of examples; SIM and MIN do not [12]. Bayes also successfully predicts how fast these similarity gradients converge to the most specific consistent rule. Convergence is quite slow in this domain ($n \sim 50$) because the hypothesis space consists of densely overlapping subsets – axis-parallel rectangles – much like the interval hypotheses in the Class III number tasks.

Another experiment engaged a word-learning task, using photographs of real objects as stimuli and a cover story of learning a new language [11]. On each trial, subjects saw either one example of a novel word (e.g., a toy animal labeled with "Here is a blicket."), or three examples at one of three different levels of specificity: subordinate (e.g., 3 dalmatians labeled with "Here are three blickets."), basic (e.g., 3 dogs), or superordinate (e.g., 3 animals). They then were asked to pick the other instances of that concept from a set of 24 test objects, containing matches to the example(s) at all levels (e.g., other dalmatians, dogs, animals) as well as many non-matching objects. Figure 2 shows data and predictions for all three models. Similarity-like generalization given one example rapidly converged to the most specific rule after only three examples were observed, just as in the number task (Classes I and II) but in contrast to the axis-parallel rectangle task or the Class III num-

ber tasks, where similarity-like responding was still the norm after three or four examples. For modeling purposes, a hypothesis space was constructed from a hierarchical clustering of subjects' similarity judgments (augmented by an a priori preference for basic-level concepts) [11]. The Bayesian model successfully predicts rapid convergence from a similarity gradient to the minimal rule, because the smallest hypothesis consistent with each example set is significantly smaller than the next-best competitor (e.g., "dogs" is significantly smaller than "dogs and cats", just as with "multiples of ten" vs. "multiples of five"). Bayes fits the full data extremely well ($R^2 = 0.98$); by comparison, SIM ($R^2 = 0.83$) successfully accounts for only the $n = 1$ trials and MIN ($R^2 = 0.76$), the $n = 3$ trials.

In conclusion, a Bayesian framework is able to account for both rule- and similarity-like modes of generalization, as well as the dynamics of transitions between these modes, across several quite different domains of concept learning. The key features of the Bayesian model are *hypothesis averaging* and the *size principle*. The former allows either rule-like or similarity-like behavior depending on the uncertainty in the posterior probability. The latter determines this uncertainty as a function of the number and distribution of examples and the structure of the learner's hypothesis space. With *sparsely overlapping* hypotheses – i.e., the most specific hypothesis consistent with the examples is much smaller than its nearest competitors – convergence to a single rule occurs rapidly, after just a few examples. With *densely overlapping* hypotheses – i.e., many consistent hypotheses of comparable size – convergence to a single rule occurs much more slowly, and a gradient of similarity is the norm after just a few examples. Importantly, the Bayesian framework does not so much obviate the distinction between rules and similarity as explain why it might be useful in understanding the brain. As Figures 1 and 2 show, special cases of Bayes corresponding to the SIM and MIN algorithms consistently account for distinct and complementary regimes of generalization. SIM, without the size principle, works best given only one example or densely overlapping hypotheses, when Equation 1 does not generate large differences in likelihood. MIN, without hypothesis averaging, works best given many examples or sparsely overlapping hypotheses, when the most specific hypothesis dominates the sum over $\mathcal{H}$ in Equation 2. In light of recent brain-imaging studies dissociating rule- and exemplar-based processing [8], the Bayesian theory may best be thought of as a computational-level account of concept learning, with multiple subprocesses – perhaps subserving SIM and MIN – implemented in distinct neural circuits. I hope to explore this possibility in future work.

## References

[1] M. Erickson & J. Kruschke (1998). Rules and exemplars in category learning. *JEP: General* **127**, 107-140.
[2] D. Haussler, M. Kearns, & R. Schapire (1994). Bounds on the sample complexity of Bayesian learning using information theory and the VC-dimension. *Machine Learning* **14**, 83-113.
[3] T. Mitchell (1997). *Machine Learning*. McGraw-Hill.
[4] R. Nosofsky & T. Palmeri (1998). A rule-plus-exception model for classifying objects in continuous-dimension spaces. *Psychonomic Bull. & Rev.* **5**, 345-369.
[5] R. Shepard (1987). Towards a universal law of generalization for psychological science. *Science* **237**, 1317-1323.
[6] R. Shepard & P. Arabie (1979). Additive clustering: Representation of similarities as combinations of discrete overlapping properties. *Psych. Rev.* **86**, 87-123.
[7] S. Sloman & L. Rips (1998). *Similarity and Symbols in Human Thinking*. MIT Press.
[8] E. Smith, A. Patalano & J. Jonides (1998). Alternative strategies of categorization. In [6].
[9] E. Smith & S. Sloman (1994). Similarity- vs. rule-based categorization. *Mem. & Cog.* **22**, 377.
[10] J. Tenenbaum (1996). Learning the structure of similarity. *NIPS 8*.
[11] J. Tenenbaum (1999). *A Bayesian Framework for Concept Learning*. Ph. D. Thesis, MIT.
[12] J. Tenenbaum (1999). Bayesian modeling of human concept learning. *NIPS 11*.


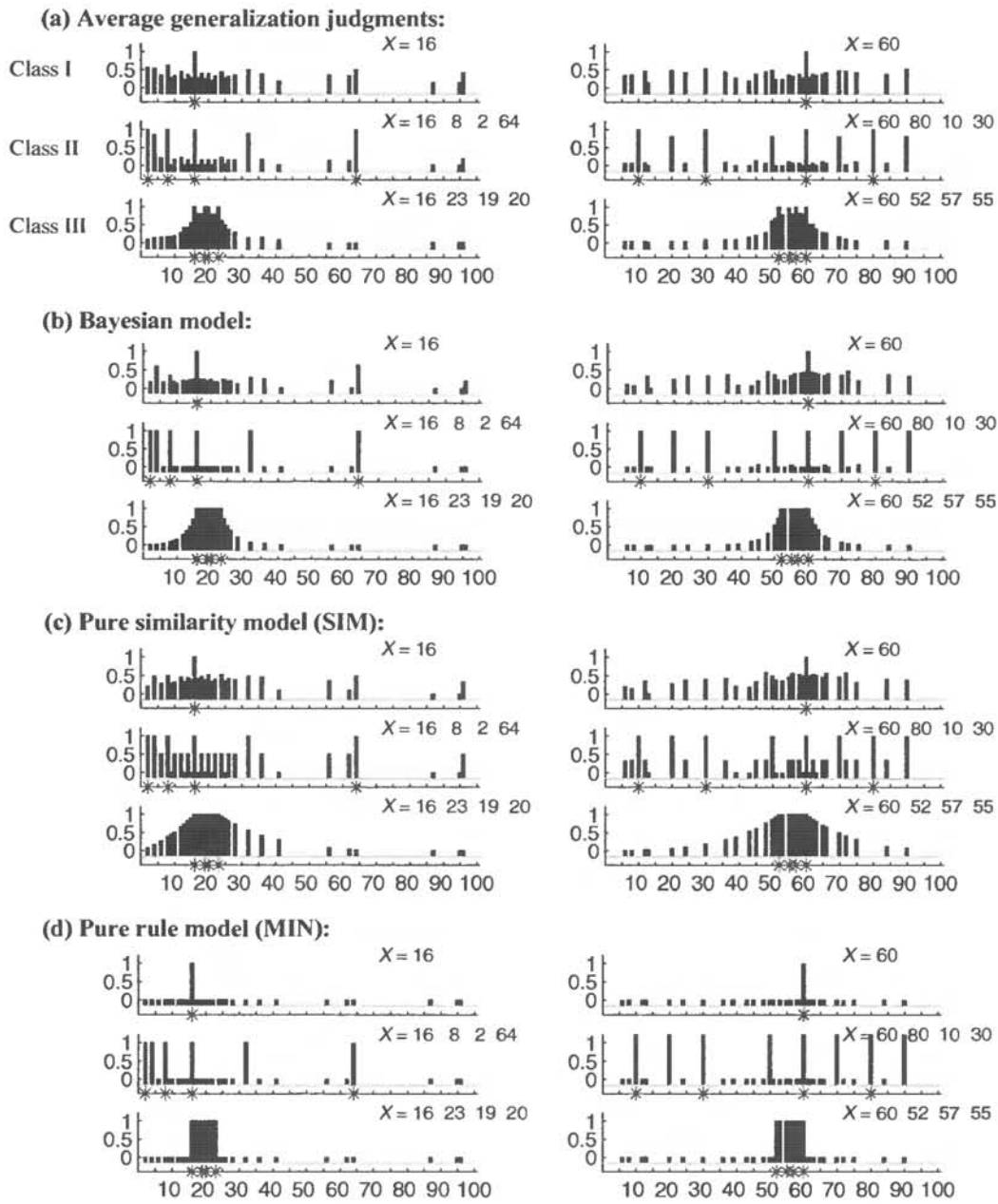

**Figure 1:** Data and model predictions for the number concept task.

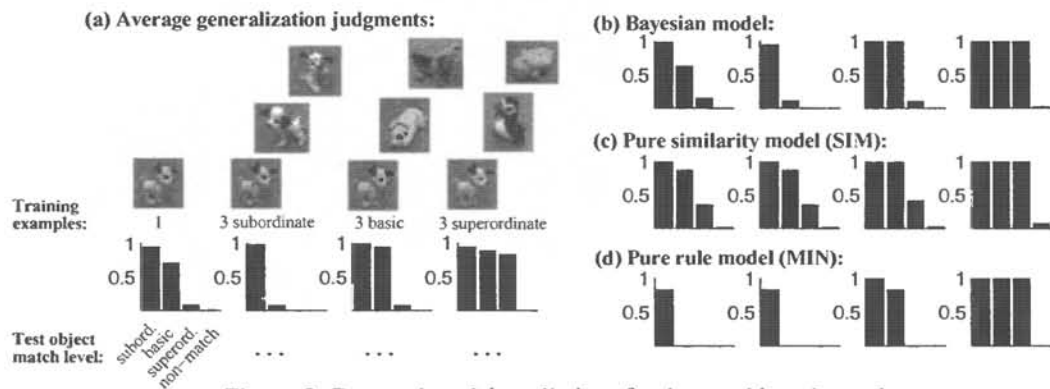

**Figure 2:** Data and model predictions for the word learning task.